# Simulation of Optimal Movements Using the Minimum-Muscle-Tension-Change Model.

**Menashe Dornay**[*]       **Yoji Uno**[**]       **Mitsuo Kawato**[*]       **Ryoji Suzuki**[**]

[*]Cognitive Processes Department, ATR Auditory and Visual Perception Research Laboratories, Sanpeidani, Inuidani, Seika-Cho, Soraku-Gun, Kyoto 619-02 Japan.

[**]Department of Mathematical Engineering and Information Physics, Faculty of Engineering, University of Tokyo, Hongo, Bunkyo-ku, Tokyo, 113 Japan.

## Abstract

This work discusses various optimization techniques which were proposed in models for controlling arm movements. In particular, the minimum-muscle-tension-change model is investigated. A dynamic simulator of the monkey's arm, including seventeen single and double joint muscles, is utilized to generate horizontal hand movements. The hand trajectories produced by this algorithm are discussed.

## 1   INTRODUCTION

To perform a voluntary hand movement, the primate nervous system must solve the following problems: (A) Which *trajectory* (hand path and velocity) should be used while moving the hand from the initial to the desired position. (B) What muscle forces should be generated. Those two problems are termed "ill-posed" because they can be solved in an infinite number of ways. The interesting question to us is: what strategy does the nervous system use while choosing a specific solution for these problems ? The chosen solutions must comply with the known experimental data: Human and monkey's free horizontal multi-joint hand movements have straight or gently curved paths. The hand velocity profiles are always roughly bell shaped (Bizzi & Abend 1986).

## 1.1    THE MINIMUM-JERK MODEL

Flash and Hogan (1985) proposed that a *global kinematic optimization* approach, the *minimum-jerk model*, defines a solution for the trajectory determination problem (problem A). Using this strategy, the nervous system is choosing the (unique) smoothest trajectory of the hand for any horizontal movement, without having to deal with the structure or dynamics of the arm. The minimum-jerk model produces reasonable approximations for hand trajectories in unconstrained point to point movements in the horizontal plane in front of the body (Flash & Hogan 1985; Morasso 1981; Uno et al. 1989a). It fails to describe, however, some important experimental findings for human arm movements (Uno et al. 1989a).

## 1.2    THE EQUILIBRIUM-TRAJECTORY HYPOTHESIS

According to the *equilibrium-trajectory hypothesis* (Feldman 1966), the nervous system generates movements by a gradual change in the equilibrium posture of the hand: at all times during the execution of a movement the muscle forces defines a stable posture which acts as a point of attraction in the configurational space of the limb. The actual hand movement is the *realized trajectory*. The realized hand trajectory is usually different from the attracting pre-planned *virtual trajectory* (Hogan 1984). Simulations by Flash (1987), have suggested that realistic multi-joint arm movements at moderate speed can be generated by moving the hand equilibrium position along a pre-planned *minimum-jerk* virtual trajectory. The interactions of the dynamic properties of the arm and the attracting virtual trajectory create together the actual realized trajectory. Flash did not suggest a solution to problem B.

A *static local optimization algorithm* related to the equilibrium-trajectory hypothesis and called *backdriving* was proposed by Mussa-Ivaldi et al. (1991). This algorithm can be used to solve problem B only after the virtual trajectory is known. The virtual trajectory is not necessarily a minimum-jerk trajectory. Driving the arm from a current equilibrium position to the next one on the virtual trajectory is performed by two steps: 1) simulate a passive displacement of the arm to the new position and 2) update the muscle forces so as to eliminate the induced hand force. A unique active change (step 2) is chosen by finding these muscle forces which minimize the change in the potential energy stored in the muscles. Using a static model of the monkey's arm, the first author has analyzed this **sequential computational approach**, including a solution for both the trajectory determination (A) and the muscle forces (B) problems (Dornay 1990, 1991a, 1991b).

The equilibrium-trajectory hypothesis which is using the minimum-jerk model was criticized by Katayama and Kawato (in preparation). According to their recent findings, the values of the dynamic stiffness used by Flash (1987) are too high to be realistic. They have found that a very complex virtual trajectory, completely different from the one predicted by the minimum-jerk model, is needed for coding realistic hand movements.

## 2    GLOBAL DYNAMIC OPTIMIZATIONS

A set of *global dynamic optimizations* have been proposed by Uno et al. (1989a, 1989b). Uno et al. suggested that the dynamic properties of the arm must be considered by any algorithm for controlling hand movements. They also proposed that the hand trajectory and the motor commands (joint torques, muscle tensions, etc.,) are computed **in parallel**.

### 2.1    THE MINIMUM-TORQUE-CHANGE MODEL

Uno et al. (1989a) have proposed the *minimum-torque-change model*. The model proposes that the hand trajectory and the joint torques are determined simultaneously, while the algorithm minimizes globally the rate of change of the joint torques. The minimum-torque-change model was criticized by Flash (1990), saying that the rotary inertia used was not realistic. If Flash's inertia values are used then the hand path predicted by the minimum-torque-change model is curved (Flash 1990).

### 2.2    THE MINIMUM-MUSCLE-TENSION-CHANGE MODEL

The *minimum-muscle-tension-change model* (Uno et al. 1989b, Dornay et al. 1991) is a parallel dynamic optimization approach in which the trajectory determination problem (A) and the muscle force generation problem (B) are solved simultaneously. No explicit trajectory is imposed on the hand, but that it must reach the final desired state (position, velocity, etc.) in a pre-specified time. The numerical solution used is a "penalty" method, in which the controller minimizes globally by iterations an energy function $E$ :

$$E = \varepsilon \left( E_D + \lambda E_s \right) \qquad (1)$$

$E$ is the energy that must be minimized in iterations. $E_D$ is a collection of hard constraints, like, for example that the hand must reach the desired position at the specified time. $E_s$ is a smoothness constraint, like the minimum-muscle-tension-change model. $\lambda$ is a regularization function, that needs to become smaller and smaller as the number of iterations increases. This is a key point because the hard constraints must be strictly satisfied at the end of the iterative process. $\varepsilon$ is a small rate term. The smoothness constraint $E_s$ , is the minimum-muscle-tension-change model, defined as:

$$E_S = 0.5 \int_{t_0}^{t_{fin}} \sum_{i=1}^{n} \left( df_i / dt \right)^2 dt \qquad (2)$$

$f_i$ is the tension of muscle i, $n$ is the total number of muscles, $t_0$ is the initial time and $t_{fin}$ is the final time of the movement.

Preliminary studies have shown (Uno et al. 1989b) that the minimum-muscle-tension-change model can simulate reasonable hand movements.

## 3  THE MONKEY'S ARM MODEL

The model used was recently described (Dornay 1991a; Dornay et al. 1991). It is based on anatomical study using the Rhesus monkey. Attachments of 17 shoulder, elbow and double joint muscles were marked on the skeleton. The skeleton was cleaned and reassembled to a natural configuration of a monkey during horizontal arm movements (Fig. 1). X-ray analysis was used to create a simplified horizontal model of the arm (Fig. 1). Effective origins and insertions of the muscles were estimated by computer simulations to ensure the postural stability of the hand at equilibrium (Dornay 1991a). The simplified dynamic model used in this study is described in Dornay et al. (1991).

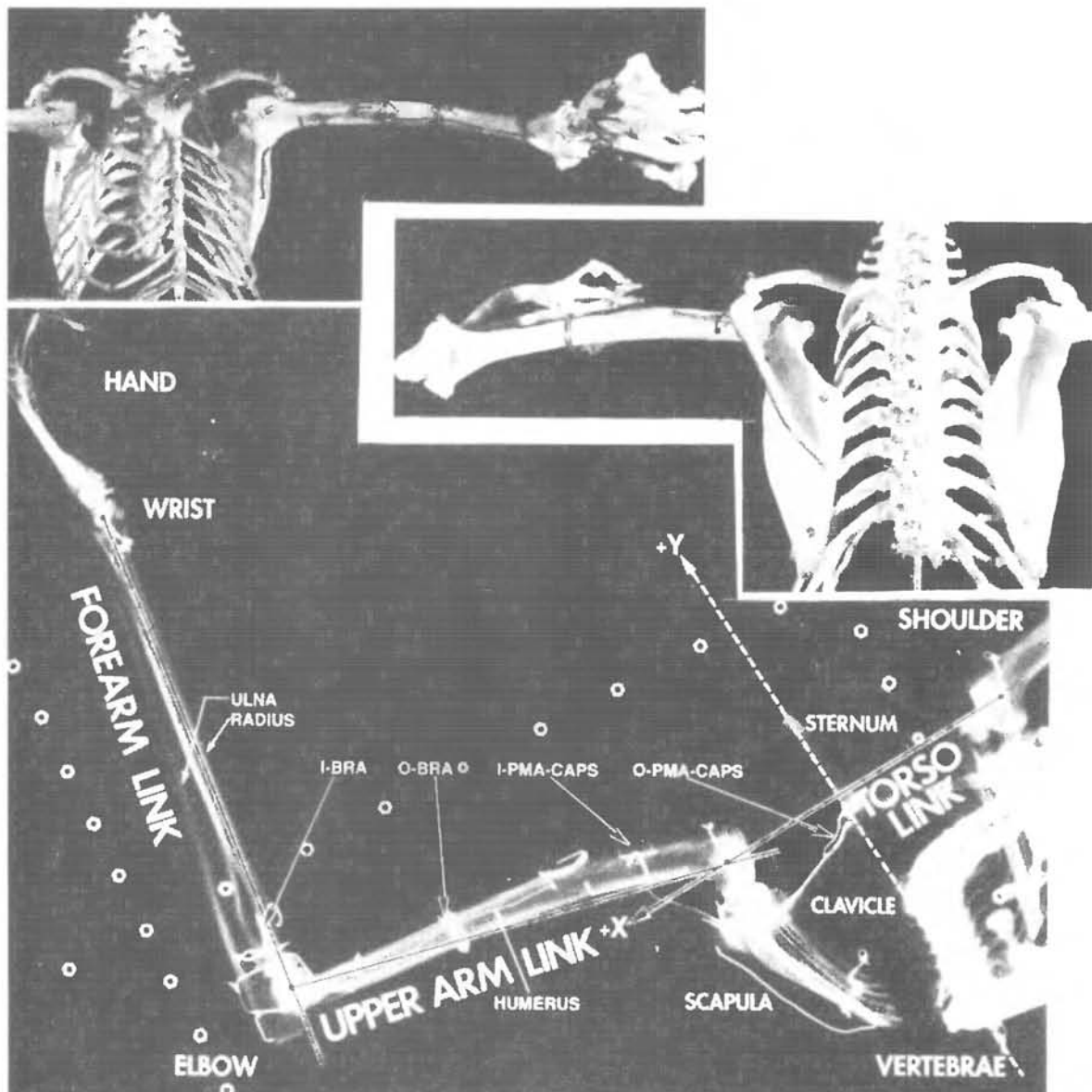

Figure 1: The Monkey's Arm Model. Top left is a ventral view of the skeleton. Middle right is a dorsal view. The bottom shows a top-down X-ray projection of the skeleton, with the axes marked on it. The photos were taken by Mr. H.S. Hall, MIT.

## 4    THE BEHAVIORAL TASK

We tried to simulate the horizontal arm movements reported by Uno et al. (1989a) for human subjects, using the monkey's model. Fig. 2 (left) shows a top view of the hand workspace of the monkey (light small dots). We used 7 hand positions defined by the following shoulder and elbow relative angles (in degrees): $T_1$ {14,122}; $T_2$ {67,100}; $T_3$ {75,64}; $T_4$ {63,45}; $T_5$ {35,54}; $T_6$ {-5,101} and $T_7$ {-25,45}. The joint angles used by Uno et al. (1989a) for $T_4$ and $T_7$, {77,22} and {0,0}, are out of the workspace of the monkey's hand (open circles in Fig 2, left). We approximated them by our $T_4$ and $T_7$ (filled circles). The behavioral task that we simulated using the minimum-muscle-tension-change model consisted of the 4 trajectories shown in Fig. 2 (right).

## 5    SIMULATION RESULTS

Figure 2 (right) shows the paths ($T_2$->$T_6$), ($T_3$->$T_6$), ($T_4$->$T_1$), and ($T_7$->$T_5$). The paths $T_2$->$T_6$, $T_3$->$T_6$ and $T_7$->$T_5$ are slightly convex. Slightly convex paths for $T_2$->$T_6$ were reported in human movements by Flash (1987), Uno et al. (1989a) and Morasso (1981). Human $T_3$->$T_6$ paths have a small tendency to be slightly convex (Uno et al. 1989a; Flash (1987). In our simulations, $T_2$->$T_6$ and $T_3$->$T_6$ have slightly larger curvatures than those reported in humans. Human large movements from the side of the body to the front of the body similar to our $T_7$->$T_5$ were reported by Uno et al. (1989a). The path of these movements is convex and similar to our simulation results. The simulated path of $T_4$->$T_1$ is slightly curved to the left and then to the right, but roughly straight. The human's $T_4$->$T_1$ paths look slightly straighter than in our simulations (Uno et al. 1989a; Flash 1987).

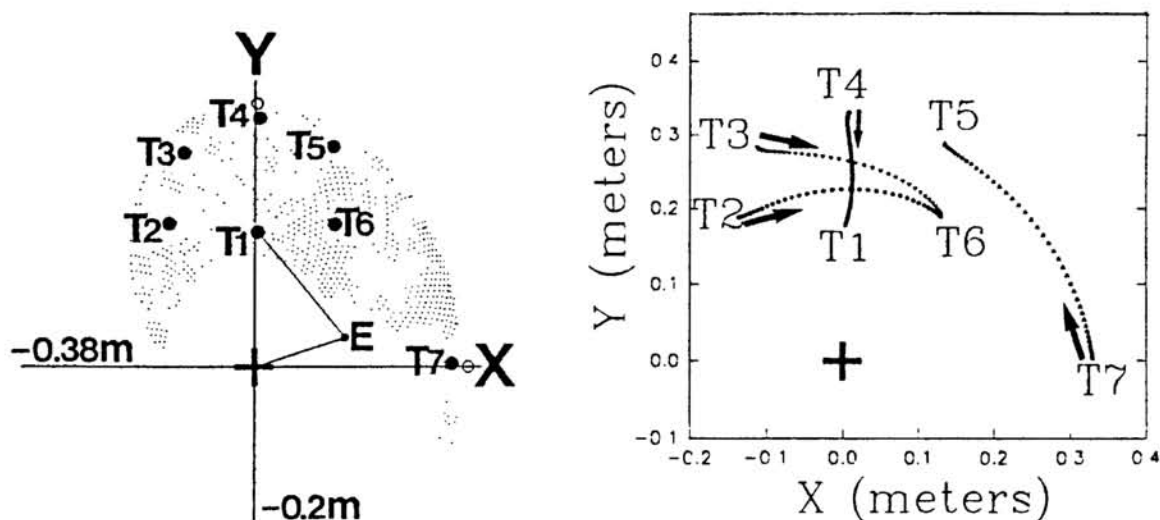

Figure 2. The Behavioral Task. The left side shows the hand workspace (small dots). The shoulder position and origin of coordinates (0,0) is marked by **+**. The elbow location when the hand is on position $T_1$ is marked by **E**. The right side shows 4 hand paths simulated by the minimum-muscle-tension-change model. Arrows indicate the directions of the movements.

Fig. 3 shows the corresponding simulated hand velocities. The velocity profiles have a single peak and are roughly bell shaped, like those reported for human subjects. The left side of the velocity profile of $T_4$->$T_1$ looks slightly irregular.

The hand trajectories simulated here are in general closer to human data than those reported by us in the past (Dornay et al. 1991). In the current study we used a much slower protocol for reducing $\lambda$ than in the previous study, and we think that we are closer now to the optimal solution of the numerical calculation than in the previous study. Indeed, the hand velocity profiles and muscle tension profiles look smoother here than in the previous study. It is in general very difficult to guarantee that the optimal solution is achieved, unless an unpractical large number of iterations is used. Fig. 4 (top,left) shows the way $E_D$ and $E_S$ of equation 1 are changing as a function $\lambda$ for the trajectory $T_7$->$T_5$. Ideally, both should reach a plato when the optimal solution is reached. The muscle tensions simulated for $T_7$->$T_5$ are shown in Fig. 4. They look quite smooth.

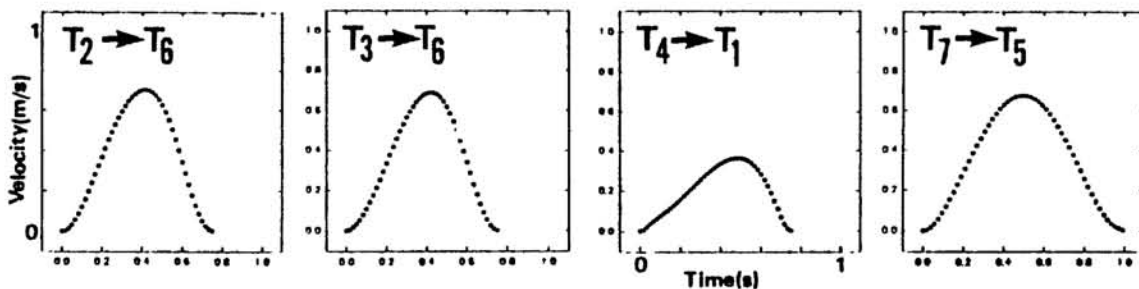

Figure 3. The Hand Tangential Velocity.

# 6   DISCUSSION

Various control strategies have been proposed to explain the roughly straight hand trajectory shown by primates in planar reaching movements. The minimum-jerk model (Flash & Hogan 1985) takes into account only the desired hand movement, and completely ignores the dynamic properties of the arm. This simplified approach is a good approximation for many movements, but cannot explain some experimental evidence (Uno et al. 1989a). A more demanding approach, the minimum-torque-change model (Uno et al. 1989a), takes into account the dynamics of the arm, but emphasizes only the torques at the joints, and completely ignores the properties of the muscles. This model was criticized to produce unrealistic hand trajectories when proper inertia values are used (Flash 1990). A third and more complicated model is the minimum-muscle-tension-change model (Uno et al. 1989b, Dornay et al. 1991). The minimum-muscle-tension-change model was shown here to produce gently curved hand movements, which although not identical, are quite close to the primate behavior. In the current study the initial and final tensions of the muscles were assumed to be zero. This is not a realistic assumption since even a static hand at an equilibrium is expected to have some stiffness. Using the minimum-muscle-tension-change model with non-zero initial and final muscle tensions is a logical

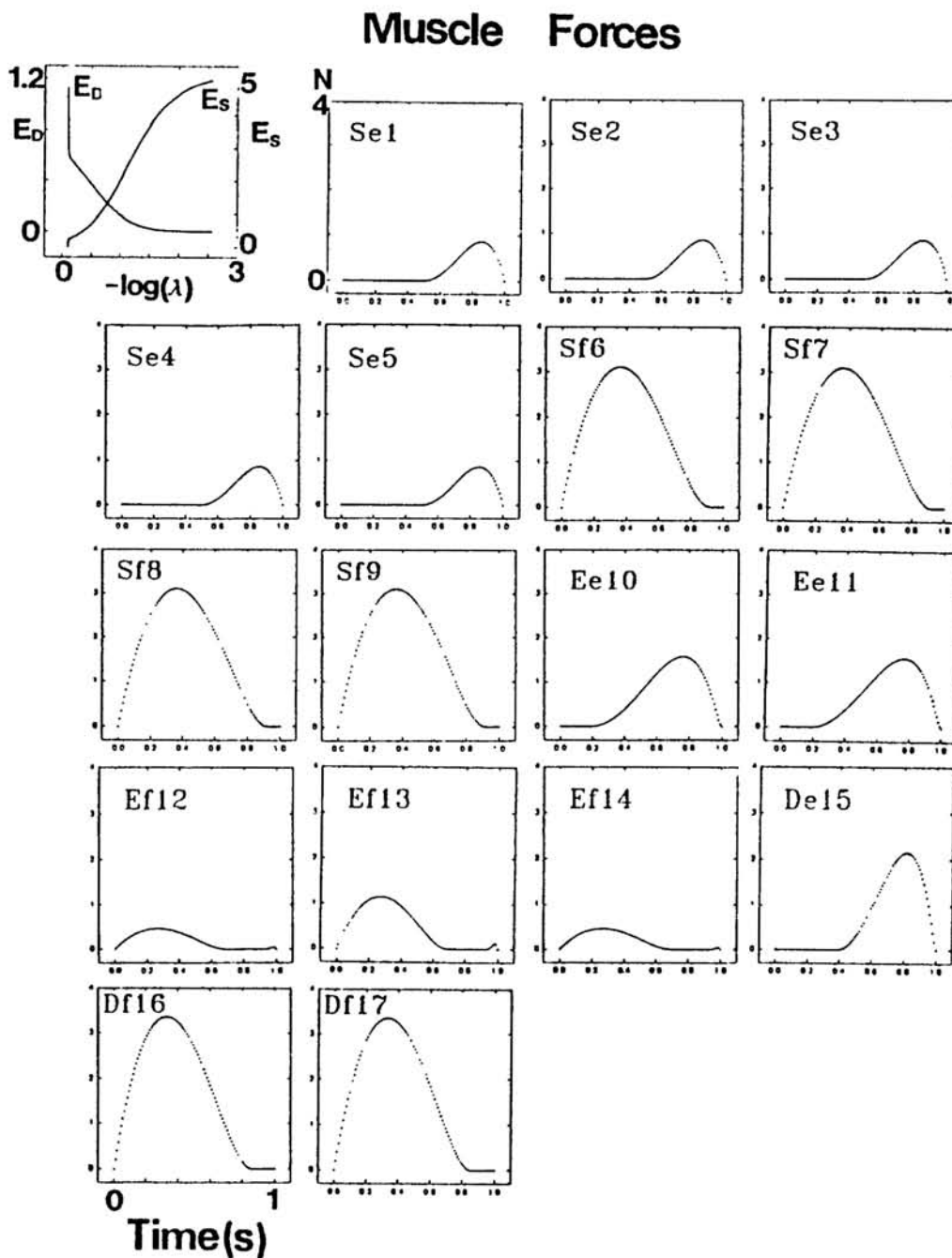

Figure 4. Numerical Analysis and Muscle Tensions For $T_7$->$T_5$. S=shoulder, E=elbow, D=double-joint muscle, e=extensor, f=flexor.

study which we intend to test in the near future. Still, the minimum-muscle-tension-change model considers only the muscle moment-arms ($\mu$) and momvels ($\partial\mu/\partial\theta$) and ignores the muscle length-tension curves. A more complicated model which we are studying now is the minimum-motor-command-change model, which includes the length-tension curves.

**Acknowledgements**

M. Dornay and M. Kawato would like to thank Drs. K. Nakane and E. Yodogawa, ATR, for their valuable help and support. Preparation of the paper was supported by Human Frontier Science Program grant to M. Kawato.

**References**

1    E Bizzi & WK Abend (1986) Control of multijoint movements. In M.J. Cohen and F. Strumwasser (Eds.) *Comparative Neurobiology: Modes of Communication in the Nervous System*, John Wiley & Sons, pp. 255-277

2    M Dornay (1990) Control of movement and the postural stability of the monkey's arm. *Proc. 3rd International Symposium on Bioelectronic and Molecular Electronic Devices*, Kobe, Japan, December 18-20, pp. 101-102

3    M Dornay (1991a) Static analysis of posture and movement, using a 17-muscle model of the monkey's arm. *ATR Technical Report TR-A-0109*

4    M Dornay (1991b) Control of movement, postural stability, and muscle angular stiffness. *Proc. IEEE Systems, Man and Cybernetics*, Virginia, USA, pp. 1373-1379

5    M Dornay, Y Uno, M Kawato & R Suzuki (1991) Simulation of optimal movements using a 17-muscle model of the monkey's arm. *Proc. SICE 30th Annual Conference, ES-1-4*, July 17-19, Yonezawam Japan, pp. 919-922

6    AG Feldman (1966) Functional tuning of the nervous system with control of movement or maintenance of a steady posture. *Biophysics*, 11, pp. 766-775

7    T Flash & N Hogan (1985) The coordination of arm movements: an experimentally confirmed mathematical model. *J. Neurosci.*, 5, pp. 1688-1703

8    T Flash (1987) The control of hand equilibrium trajectories in multi-joint arm movements. *Biol. Cybern.*, 57, pp. 257-274

9    T Flash (1990) The organization of human arm trajectory control. In J. Winters and S. Woo (Eds.) *Multiple muscle systems: Biomechanics and movement organization*, Springer-Verlag, pp. 282-301

10   N Hogan (1984) An organizing principle for a class of voluntary movements. *J. Neurosci.*, 4, pp. 2745-2754

11   P Morasso (1981) Spatial control of arm movements. *Experimental Brain Research*, 42, pp. 223-227

12   FA Mussa-Ivaldi, P Morasso, N Hogan & E Bizzi (1991) Network models of motor systems with many degrees of freedom. In M.D. Fraser (Ed.) *Advances in control networks and large scale parallel distributed processing models*, Albex Publ. Corp.

13   Y Uno, M Kawato & R Suzuki (1989a) Formation and control of optimal trajectory in human multijoint arm movement - minimum-torque-change model. *Biol. Cybern.*, 61, pp. 89-101

14   Y Uno, R Suzuki & M Kawato (1989b) Minimum muscle-tension change model which reproduces human arm movement. *Proceedings of the 4th Symposium on Biological and Physiological Engineering*, pp. 299-302, (in Japanese)
